# Sparse Overcomplete Latent Variable Decomposition of Counts Data

**Madhusudana Shashanka**
Mars, Incorporated
Hackettstown, NJ
shashanka@cns.bu.edu

**Bhiksha Raj**
Mitsubishi Electric Research Labs
Cambridge, MA
bhiksha@merl.com

**Paris Smaragdis**
Adobe Systems
Newton, MA
paris@adobe.com

## Abstract

An important problem in many fields is the analysis of counts data to extract meaningful latent components. Methods like Probabilistic Latent Semantic Analysis (PLSA) and Latent Dirichlet Allocation (LDA) have been proposed for this purpose. However, they are limited in the number of components they can extract and lack an explicit provision to control the "expressiveness" of the extracted components. In this paper, we present a learning formulation to address these limitations by employing the notion of sparsity. We start with the PLSA framework and use an entropic prior in a maximum a posteriori formulation to enforce sparsity. We show that this allows the extraction of overcomplete sets of latent components which better characterize the data. We present experimental evidence of the utility of such representations.

## 1 Introduction

A frequently encountered problem in many fields is the analysis of histogram data to extract meaningful *latent* factors from it. For text analysis where the data represent counts of word occurrences from a collection of documents, popular techniques available include Probabilistic Latent Semantic Analysis (PLSA; [6]) and Latent Dirichlet Allocation (LDA; [2]). These methods extract components that can be interpreted as *topics* characterizing the corpus of documents. Although they are primarily motivated by the analysis of text, these methods can be applied to analyze arbitrary count data. For example, images can be interpreted as histograms of multiple draws of pixels, where each draw corresponds to a "quantum of intensity". PLSA allows us to express distributions that underlie such count data as mixtures of latent components. Extensions to PLSA include methods that attempt to model how these components co-occur (eg. LDA, Correlated Topic Model [1]).

One of the main limitations of these models is related to the number of components they can extract. Realistically, it may be expected that the number of latent components in the process underlying any dataset is unrestricted. However, the number of components that can be discovered by LDA or PLSA is restricted by the cardinality of the data, *e.g.* by the vocabulary of the documents, or the number of pixels of the image analyzed. Any analysis that attempts to find an *overcomplete* set of a larger number of components encounters the problem of indeterminacy and is liable to result in meaningless or trivial solutions. The second limitation of the models is related to the "expressiveness" of the extracted components *i.e.* the information content in them. Although the methods *aim* to find "meaningful" latent components, they do not actually provide any control over the information content in the components.

In this paper, we present a learning formulation that addresses both these limitations by employing the notion of *sparsity*. *Sparse coding* refers to a representational scheme where, of a set of components that may be combined to compose data, only a small number are combined to represent any particular instance of the data (although the specific set of components may change from instance to

instance). In our problem, this translates to permitting the generating process to have an unrestricted number of latent components, but requiring that only a small number of them contribute to the composition of the histogram represented by any data instance. In other words, the latent components must be learned such that the *mixture weights* with which they are combined to generate any data have low entropy – a set with low entropy implies that only a few mixture weight terms are significant. This addresses both the limitations. Firstly, it largely eliminates the problem of indeterminacy permitting us to learn an unrestricted number of latent components. Secondly, estimation of low entropy mixture weights forces more information on to the latent components, thereby making them more expressive.

The basic formulation we use to extract latent components is similar to PLSA. We use an *entropic prior* to manipulate the entropy of the mixture weights. We formulate the problem in a *maximum a posteriori* framework and derive inference algorithms. We use an artificial dataset to illustrate the effects of sparsity on the model. We show through simulations that sparsity can lead to components that are more representative of the true nature of the data compared to conventional maximum likelihood learning. We demonstrate through experiments on images that the latent components learned in this manner are more informative enabling us to predict unobserved data. We also demonstrate that they are more discriminative than those learned using regular maximum likelihood methods. We then present conclusions and avenues for future work.

## 2 Latent Variable Decomposition

Consider an $F \times N$ count matrix $\mathbf{V}$. We will consider each column of $\mathbf{V}$ to be the histogram of an independent set of draws from an underlying multinomial distribution over $F$ discrete values. Each column of $\mathbf{V}$ thus represents counts in a unique data set. $V_{fn}$, the $f^{\text{th}}$ row entry of $\mathbf{V}_n$, the $n^{\text{th}}$ column of $\mathbf{V}$, represents the count of $f$ (or the $f^{\text{th}}$ discrete symbol that may be generated by the multinomial) in the $n^{\text{th}}$ data set. For example, if the columns of $\mathbf{V}$ represent word count vectors for a collection of documents, $V_{fn}$ would be the count of the $f^{\text{th}}$ word of the vocabulary in the $n^{\text{th}}$ document in the collection.

We model all data as having been generated by a process that is characterized by a set of *latent* probability distributions that, although not directly observed, combine to compose the distribution of any data set. We represent the probability of drawing $f$ from the $z^{\text{th}}$ latent distribution by $P(f|z)$, where $z$ is a latent variable. To generate any data set, the latent distributions $P(f|z)$ are combined in proportions that are specific to that set. Thus, each histogram (column) in $\mathbf{V}$ is the outcome of draws from a distribution that is a column-specific composition of $P(f|z)$. We can define the distribution underlying the $n^{\text{th}}$ column of $\mathbf{V}$ as

$$P_n(f) = \sum_z P(f|z)P_n(z), \tag{1}$$

where $P_n(f)$ represents the probability of drawing $f$ in the $n^{\text{th}}$ data set in $\mathbf{V}$, and $P_n(z)$ is the mixing proportion signifying the contribution of $P(f|z)$ towards $P_n(f)$.

Equation 1 is functionally identical to that used for Probabilistic Latent Semantic Analysis of text data [6][1]: if the columns $\mathbf{V}_n$ of $\mathbf{V}$ represent word count vectors for documents, $P(f|z)$ represents the $z^{\text{th}}$ latent *topic* in the documents. Analogous interpretations may be proposed for other types of data as well. For example, if each column of $\mathbf{V}$ represents one of a collection of images (each of which has been unraveled into a column vector), the $P(f|z)$'s would represent the latent "bases" that compose all images in the collection. In maintaining this latter analogy, we will henceforth refer to $P(f|z)$ as the *basis* distributions for the process.

Geometrically, the *normalized* columns of $\mathbf{V}$ (obtained by scaling the entries of $\mathbf{V}_n$ to sum to 1.0), $\bar{\mathbf{V}}_n$, which we refer to as *data distributions*, may be viewed as $F$-dimensional vectors that lie in an $(F-1)$ simplex. The distributions $P_n(f)$ and basis distributions $P(f|z)$ are also $F$-dimensional vectors in the same simplex. The model expresses $P_n(f)$ as points within the convex hull formed by the basis distributions $P(f|z)$. The aim of the model is to determine $P(f|z)$ such that the model

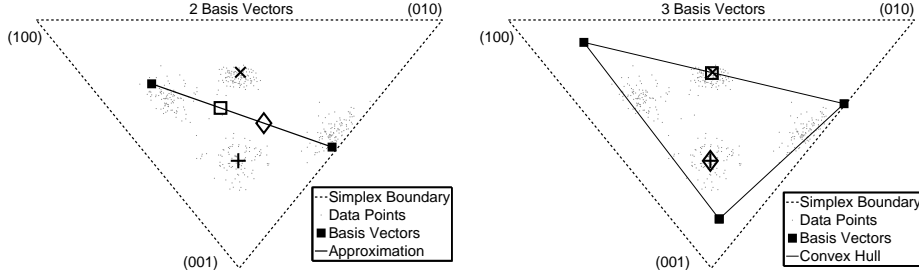

Figure 1: Illustration of the latent variable model. Panels show 3-dimensional data distributions as points within the *Standard 2-Simplex* given by $\{(001),(010),(100)\}$. The left panel shows a set of 2 Basis distributions (*compact code*) derived from the 400 data points. The right panel shows a set of 3 Basis distributions (*complete code*). The model approximates data distributions as points lying within the convex hull formed by the basis distributions. Also shown are two data points (marked by $+$ and $\times$) and their approximations by the model (respectively shown by $\diamond$ and $\square$).

$P_n(f)$ for any data distribution $\bar{\mathbf{V}}_n$ approximates it closely. Since $P_n(f)$ is constrained to lie within the simplex defined by $P(f|z)$, it can only model $\bar{\mathbf{V}}_n$ accurately if the latter also lies within the hull. Any $\bar{\mathbf{V}}_n$ that lies outside the hull is modeled with error. Thus, the objective of the model is to identify $P(f|z)$ such that they form a convex hull surrounding the data distributions. This is illustrated in Figure 1 for a synthetic data set of 400 3-dimensional data distributions.

## 2.1 Parameter Estimation

Given count matrix $\mathbf{V}$, we estimate $P(f|z)$ and $P_n(z)$ to maximize the likelihood of $\mathbf{V}$. This can be done through iterations of equations derived using the Expectation Maximization (EM) algorithm:

$$P_n(z|f) = \frac{P_n(z)P(f|z)}{\sum_z P_n(z)P(f|z)}, \quad \text{and} \tag{2}$$

$$P(f|z) = \frac{\sum_n V_{fn}P_n(z|f)}{\sum_f \sum_n V_{fn}P_n(z|f)}, \qquad P_n(z) = \frac{\sum_f V_{fn}P_n(z|f)}{\sum_z \sum_f V_{fn}P_n(z|f)} \tag{3}$$

Detailed derivation is shown in supplemental material. The EM algorithm guarantees that the above multiplicative updates converge to a local optimum.

## 2.2 Latent Variable Model as Matrix Factorization

We can write the model given by equation (1) in matrix form as $\mathbf{p}_n = \mathbf{W}\mathbf{g}_n$, where $\mathbf{p}_n$ is a column vector indicating $P_n(f)$, $\mathbf{g}_n$ is a column vector indicating $P_n(z)$, and $\mathbf{W}$ is a matrix with the $(f,z)$-th element corresponding to $P(f|z)$. If we characterize $\mathbf{V}$ by $R$ basis distributions, $\mathbf{W}$ is an $F \times R$ matrix. Concatenating all column vectors $\mathbf{p}_n$ and $\mathbf{g}_n$ as matrices $\mathbf{P}$ and $\mathbf{G}$ respectively, one can write the model as $\mathbf{P} = \mathbf{W}\mathbf{G}$, where $\mathbf{G}$ is an $R \times N$ matrix. It is easy to show (as demonstrated in the supplementary material) that the maximum likelihood estimator for $P(f|z)$ and $P_n(z)$ attempts to minimize the Kullback-Leibler (KL) distance between the normalized data distribution $\mathbf{V}_n$ and $P_n(f)$, weighted by the total count in $\mathbf{V}_n$. In other words, the model of Equation (1) actually represents the decomposition

$$\mathbf{V} \approx \mathbf{W}\mathbf{G}\mathbf{D} = \mathbf{W}\mathbf{H} \tag{4}$$

where $\mathbf{D}$ is an $N \times N$ diagonal matrix, whose $n^{\text{th}}$ diagonal element is the total number of counts in $\mathbf{V}_n$ and $\mathbf{H} = \mathbf{G}\mathbf{D}$. The astute reader might recognize the decomposition of equation (4) as Non-negative matrix factorization (NMF; [8]). In fact equations (2) and (3) can be shown to be equivalent to one of the standard update rules for NMF.

Representing the decomposition in matrix form immediately reveals one of the shortcomings of the basic model. If $R$, the number of basis distributions, is equal to $F$, then a trivial solution exists that achieves perfect decomposition: $\mathbf{W} = \mathbf{I}$; $\mathbf{H} = \mathbf{V}$, where $\mathbf{I}$ is the identity matrix (although the algorithm may not always arrive at this solution). However, this solution is no longer of any utility to us since our aim is to derive basis distributions that are characteristic of the data, whereas the

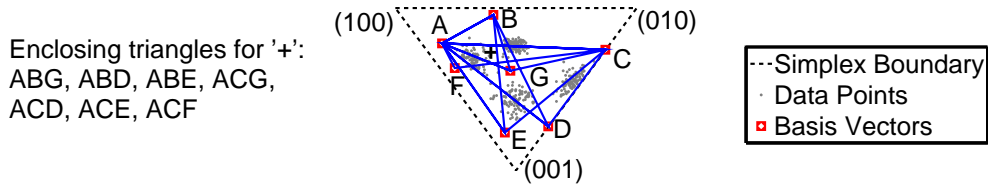

Enclosing triangles for '+':
ABG, ABD, ABE, ACG,
ACD, ACE, ACF

Figure 2: Illustration of the effect of sparsifying $\mathbf{H}$ on the dataset shown in Figure 1. A-G represent 7 basis distributions. The '+' represents a typical data point. It can be accurately represented by any set of three or more bases that form an enclosing polygon and there are many such polygons. However, if we restrict the number of bases used to enclose '+' to be minimized, only the 7 enclosing triangles shown remain as valid solutions. By further imposing the restriction that the entropy of the mixture weights with which the bases (corners) must be combined to represent '+' must be minimum, only one triangle is obtained as the unique optimal enclosure.

columns of $\mathbf{W}$ in this trivial solution are not specific to any data, but represent the dimensions of the *space* the data lie in. For *overcomplete* decompositions where $R > F$, the solution becomes indeterminate – multiple perfect decompositions are possible.

The indeterminacy of the overcomplete decomposition can, however, be greatly reduced by imposing a restriction that the approximation for any $\bar{\mathbf{V}}_n$ must employ minimum number of basis distributions required. By further imposing the constraint that the *entropy* of $\mathbf{g}_n$ must be minimized, the indeterminacy of the solution can often be eliminated as illustrated by Figure 2. This principle, which is related to the concept of *sparse coding* [5], is what we will use to derive overcomplete sets of basis distributions for the data.

## 3 Sparsity in the Latent Variable Model

Sparse coding refers to a representational scheme where, of a set of components that may be combined to compose data, only a small number are combined to represent any particular input. In the context of basis decompositions, the goal of sparse coding is to find a set of bases for any data set such that the mixture weights with which the bases are combined to compose any data are sparse. Different metrics have been used to quantify the sparsity of the mixture weights in the literature. Some approaches minimize variants of the $L_p$ norm of the mixture weights (eg. [7]) while other approaches minimize various approximations of the entropy of the mixture weights.

In our approach, we use entropy as a measure of sparsity. We use the *entropic prior*, which has been used in the *maximum entropy* literature (see [9]) to manipulate entropy. Given a probability distribution $\boldsymbol{\theta}$, the entropic prior is defined as $P_e(\boldsymbol{\theta}) \propto e^{-\alpha \mathcal{H}(\boldsymbol{\theta})}$, where $\mathcal{H}(\boldsymbol{\theta}) = -\sum_i \theta_i \log \theta_i$ is the entropy of the distribution and $\alpha$ is a weighting factor. Positive values of $\alpha$ favor distributions with lower entropies while negative values of $\alpha$ favor distributions with higher entropies. Imposing this prior during *maximum a posteriori* estimation is a way to manipulate the entropy of the distribution. The distribution $\boldsymbol{\theta}$ could correspond to the basis distributions $P(f|z)$ or the mixture weights $P_n(z)$ or both. A sparse code would correspond to having the entropic prior on $P_n(z)$ with a positive value for $\alpha$. Below, we consider the case where both the basis vectors and mixture weights have the entropic prior to keep the exposition general.

### 3.1 Parameter Estimation

We use the EM algorithm to derive the update equations. Let us examine the case where both $P(f|z)$ and $P_n(z)$ have the entropic prior. The set of parameters to be estimated is given by $\Lambda = \{P(f|z), P_n(z)\}$. The *a priori* distribution over the parameters, $P(\Lambda)$, corresponds to the entropic priors. We can write $\log P(\Lambda)$, the log-prior, as

$$\alpha \sum_z \sum_f P(f|z) \log P(f|z) + \beta \sum_n \sum_z P_n(z) \log P_n(z), \qquad (5)$$

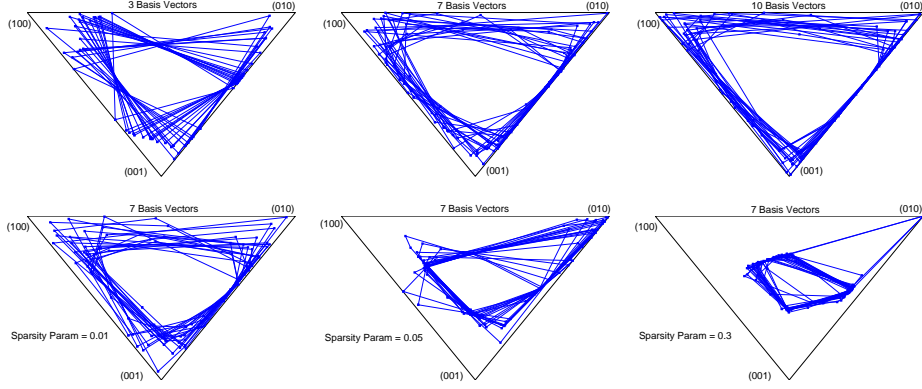

Figure 3: Illustration of the effect of sparsity on the synthetic data set from Figure 1. For visual clarity, we do not display the data points.
*Top panels: Decomposition without sparsity.* Sets of 3 (left), 7 (center), and 10 (right) basis distributions were obtained from the data without employing sparsity. In each case, 20 runs of the estimation algorithm were performed from different initial values. The convex hulls formed by the bases from each of these runs are shown in the panels from left to right. Notice that increasing the number of bases enlarges the sizes of convex hulls, none of which characterize the distribution of the data well.
*Bottom panels: Decomposition with sparsity.* The panels from left to right show the 20 sets of estimates of 7 basis distributions, for increasing values of the sparsity parameter for the mixture weights. The convex hulls quickly shrink to compactly enclose the distribution of the data.

where $\alpha$ and $\beta$ are parameters indicating the degree of sparsity desired in $P(f|z)$ and $P_n(z)$ respectively. As before, we can write the E-step as

$$P_n(z|f) = \frac{P_n(z)P(f|z)}{\sum_z P_n(z)P(f|z)}. \tag{6}$$

The M-step reduces to the equations

$$\frac{\xi}{P(f|z)} + \alpha + \alpha \log P(f|z) + \rho_z = 0, \quad \frac{\omega}{P_n(z)} + \beta + \beta \log P_n(z) + \tau_n = 0 \tag{7}$$

where we have let $\xi$ represent $\sum_n V_{fn} P_n(z|f)$, $\omega$ represent $\sum_f V_{fn} P_n(z|f)$, and $\rho_z$, $\tau_n$ are Lagrange multipliers. The above M-step equations are systems of simultaneous transcendental equations for $P(f|z)$ and $P_n(z)$. Brand [3] proposes a method to solve such equations using the Lambert $\mathcal{W}$ function [4]. It can be shown that $P(f|z)$ and $P_n(z)$ can be estimated as

$$\hat{P}(f|z) = \frac{-\xi/\alpha}{\mathcal{W}(-\xi e^{1+\rho_z/\alpha}/\alpha)}, \qquad \hat{P}_n(z) = \frac{-\omega/\beta}{\mathcal{W}(-\omega e^{1+\tau_n/\beta}/\beta)}. \tag{8}$$

Equations (7), (8) form a set of fixed-point iterations that typically converge in 2-5 iterations [3].

The final update equations are given by equation (6), and the fixed-point equation-pairs (7), (8). Details of the derivation are provided in supplemental material. Notice that the above equations reduce to the maximum likelihood updates of equations (2) and (3) when $\alpha$ and $\beta$ are set to zero. More generally, the EM algorithm aims to minimize the KL distance between the true distribution of the data and that of the model, *i.e.* it attempts to arrive at a model that conserves the entropy of the data, subject to the *a priori* constraints. Consequently, reducing entropy of the mixture weights $P_n(z)$ to obtain a sparse code results in increased entropy (information) of basis distributions $P(f|z)$.

### 3.2 Illustration of the Effect of Sparsity

The effect and utility of sparse overcomplete representations is demonstrated by Figure 3. In this example, the data (from Figure 1) have four distinct quadrilaterally located clusters. This structure cannot be accurately represented by three or fewer basis distributions, since they can, at best specify

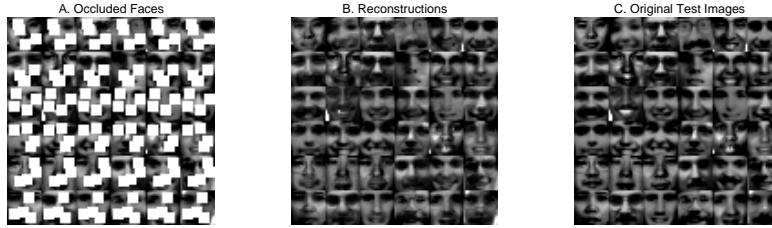

Figure 4: Application of latent variable decomposition for reconstructing faces from occluded images (*CBCL Database*). (A). Example of a random subset of 36 occluded test images. Four $6 \times 6$ patches were removed from the images in several randomly chosen configurations (corresponding to the rows). (B). Reconstructed faces from a sparse-overcomplete basis set of 1000 learned components (sparsity parameter = 0.1). (C). Original test images shown for comparison.

a triangular simplex, as demonstrated by the top left panel in the figure. Simply increasing the number of bases without constraining the sparsity of the mixture weights does not provide meaningful solutions. However, increasing the sparsity quickly results in solutions that accurately characterize the distribution of the data.

A clearer intuition is obtained when we consider the matrix form of the decomposition in Equation 4. The goal of the decomposition is often to identify a set of latent distributions that characterize the underlying *process* that generated the data $\mathbf{V}$. When no sparsity is enforced on the solution, the trivial solution $\mathbf{W} = \mathbf{I}$, $\mathbf{H} = \mathbf{V}$ is obtained at $R = F$. In this solution, the entire information in $\mathbf{V}$ is borne by $\mathbf{H}$ and the bases $\mathbf{W}$ becomes uninformative, *i.e.* they no longer contain information about the underlying process.

However, by enforcing sparsity on $\mathbf{H}$ the information $\mathbf{V}$ is transferred back to $\mathbf{W}$, and non-trivial solutions are possible for $R > F$. As $R$ increases, however, $\mathbf{W}$ become more and more data-like. At $R = N$ another trivial solution is obtained: $\mathbf{W} = \mathbf{V}$, and $\mathbf{H} = \mathbf{D}$ (*i.e.* $\mathbf{G} = \mathbf{I}$). The columns of $\mathbf{W}$ now simply represent (scaled versions) of the specific data $\mathbf{V}$ rather than the underlying process. For $R > N$ the solutions will now become indeterminate. By enforcing sparsity, we have thus increased the implicit limit on the number of bases that can be estimated without indeterminacy from the smaller dimension of $\mathbf{V}$ to the larger one.

# 4 Experimental Evaluation

We hypothesize that if the learned basis distribution are characteristic of the process that generates the data, they must not only generalize to explain new data from the process, but also enable prediction of components of the data that were not observed. Secondly, the bases for a given process must be worse at explaining data that have been generated by any other process. We test both these hypotheses below. In both experiments we utilize images, which we interpret as histograms of repeated draws of pixels, where each draw corresponds to a quantum of intensity.

## 4.1 Face Reconstruction

In this experiment we evaluate the ability of the overcomplete bases to explain new data and predict the values of unobserved components of the data. Specifically, we use it to reconstruct occluded portions of images. We used the *CBCL database* consisting of 2429 frontal view face images hand-aligned in a $19 \times 19$ grid. We preprocessed the images by linearly scaling the grayscale intensities so that pixel mean and standard deviation was 0.25, and then clipped them to the range [0, 1]. 2000 images were randomly chosen as the training set. 100 images from the remaining 429 were randomly chosen as the test set. To create occluded test images, we removed $6 \times 6$ grids in ten random configurations for 10 test faces each, resulting in 100 occluded images. We created 4 sets of test images, where each set had one, two, three or four $6 \times 6$ patches removed. Figure 4A represents the case where 4 patches were removed from each face.

In a training stage, we learned sets of $K \in \{50, 200, 500, 750, 1000\}$ basis distributions from the training data. Sparsity was not used in the compact ($R < F$) case (50 and 200 bases) and sparsity

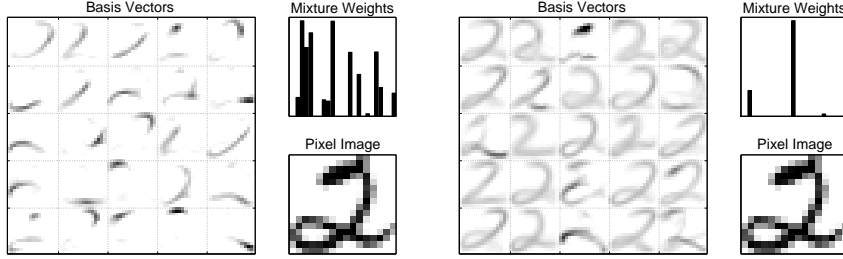

Figure 5: 25 Basis distributions (represented as images) extracted for class "2" from training data without sparsity on mixture weights (Left Panel, sparsity parameter = 0) and with sparsity on mixture weights (Right Panel, sparsity parameter = 0.2). Basis images combine in proportion to the mixture weights shown to result in the pixel images shown.

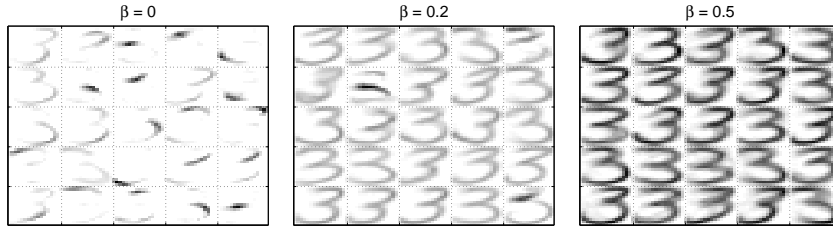

Figure 6: 25 basis distributions learned from training data for class "3" with increasing sparsity parameters on the mixture weights. The sparsity parameter was set to 0, 0.2 and 0.5 respectively. Increasing the sparsity parameter of mixture weights produces bases which are holistic representations of the input (histogram) data instead of parts-like features.

was imposed (parameter = 0.1) on the mixture weights in the overcomplete cases (500, 750 and 1000 basis vectors).

The procedure for estimating the occluded regions of the a test image has two steps. In the first step, we estimate the distribution underlying the image as a linear combination of the basis distributions. This is done by iterations of Equations 2 and 3 to estimate $P_n(z)$ (the bases $P(f|z)$, being already known, stay fixed) based only on the pixels that are observed (*i.e.* we marginalize out the occluded pixels). The combination of the bases $P(f|z)$ and the estimated $P_n(z)$ give us the overall distribution $P_n(f)$ for the image. The occluded pixel values at any pixel $f$ is estimated as the *expected* number of counts at the pixels, given by $P_n(f)(\sum_{f' \in \{F_o\}} V_{f'})/(\sum_{f' \in \{F_o\}} P_n(f'))$ where $V_f$ represents the value of the image at the $f^{\text{th}}$ pixel and $\{F_o\}$ is the set of observed pixels. Figure 4B shows the reconstructed faces for the sparse-overcomplete case of 1000 basis vectors. Figure 7A summarizes the results for all cases. Performance is measured by mean Signal-to-Noise-Ratio (SNR), where SNR for an image was computed as the ratio of the sum of squared pixel intensities of the original image to the sum of squared error between the original image pixels and the reconstruction.

## 4.2 Handwritten Digit Classification

In this experiment we evaluate the specificity of the bases to the process represented by the training data set, through a simple example of handwritten digit classification. We used the USPS Handwritten Digits database which has 1100 examples for each digit class. We randomly chose 100 examples from each class and separated them as the test set. The remaining examples were used for training. During training, separate sets of basis distributions $P^k(f|z)$ were learned for each class, where $k$ represents the index of the class. Figure 5 shows 25 bases images extracted for the digit "2". To classify any test image $v$, we attempted to compute the distribution underlying the image using the bases for each class (by estimating the mixture weights $P_v^k(z)$, keeping the bases fixed, as before). The "match" of the bases to the test instance was indicated by the likelihood $\mathcal{L}^k$ of the image computed using $P^k(f) = \sum_z P^k(f|z)P_v^k(z)$ as $\mathcal{L}^k = \sum_f v_f \log P^k(f)$. Since we expect the bases for the true class of the image to best compose it, we expect the likelihood for the correct class to be maximum. Hence, the image **v** was assigned to the class for which likelihood was the highest.

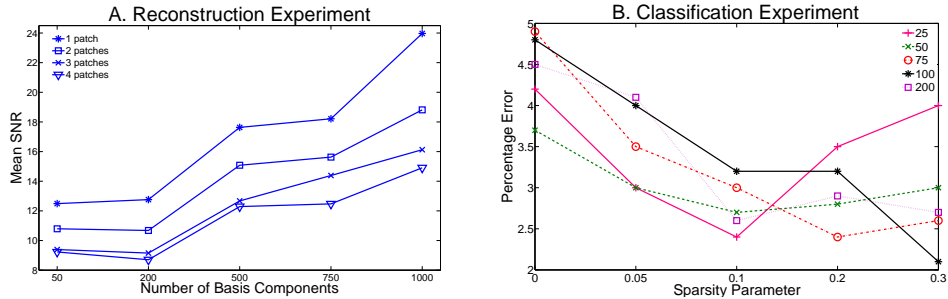

Figure 7: (A). Results of the face Reconstruction experiment. Mean SNR of the reconstructions is shown as a function of the number of basis vectors and the test case (number of deleted patches, shown in the legend). Notice that the sparse-overcomplete codes consistently perform better than the compact codes. (B). Results of the classification experiment. The legend shows number of basis distributions used. Notice that imposing sparsity almost always leads to better classification performance. In the case of 100 bases, error rate comes down by almost 50% when a sparsity parameter of 0.3 is imposed.

Results are shown in Figure 7B. As one can see, imposing sparsity improves classification performance in almost all cases. Figure 6 shows three sets of basis distributions learned for class "3" with different sparsity values on the mixture weights. As the sparsity parameter is increased, bases tend to be holistic representations of the input histograms. This is consistent with improved classification performance - as the representation of basis distributions gets more holistic, the more *unlike* they become when compared to bases of other classes. Thus, there is a lesser chance that the bases of one class can compose an image in another class, thereby improving performance.

## 5    Conclusions

In this paper, we have presented an algorithm for sparse extraction of overcomplete sets of latent distributions from histogram data. We have used entropy as a measure of sparsity and employed the entropic prior to manipulate the entropy of the estimated parameters. We showed that sparse-overcomplete components can lead to an improved characterization of data and can be used in applications such as classification and inference of missing data. We believe further improved characterization may be achieved by the imposition of additional priors that represent known or hypothesized structure in the data, and will be the focus of future research.

## Footnotes

[1]PLSA actually represents the *joint* distribution of $n$ and $f$ as $P(n, f) = P(n) \sum_z P(f|z)P(z|n)$. However the maximum likelihood estimate of $P(n)$ is simply the fraction of all observations from all data sets that occurred in the $n^{\text{th}}$ data set and does not affect the estimation of $P(f|z)$ and $P(z|n)$.

## References

[1] DM Blei and JD Lafferty. Correlated Topic Models. In *NIPS*, 2006.

[2] DM Blei, AY Ng, and MI Jordan. Latent Dirichlet Allocation. *Journal of Machine Learning Research*, 3:993–1022, 2003.

[3] ME Brand. Pattern Discovery via Entropy Minimization. In *Uncertainty 99: AISTATS 99*, 1999.

[4] RM Corless, GH Gonnet, DEG Hare, DJ Jeffrey, and DE Knuth. On the Lambert $\mathcal{W}$ Function. *Advances in Computational mathematics*, 1996.

[5] DJ Field. What is the Goal of Sensory Coding? *Neural Computation*, 1994.

[6] T Hofmann. Unsupervised Learning by Probabilistic Latent Semantic Analysis. *Machine Learning*, 42:177–196, 2001.

[7] PO Hoyer. Non-negative Matrix Factorization with Sparseness Constraints. *Journal of Machine Learning Research*, 5, 2004.

[8] DD Lee and HS Seung. Algorithms for Non-negative Matrix Factorization. In *NIPS*, 2001.

[9] J Skilling. Classic Maximum Entropy. In J Skilling, editor, *Maximum Entropy and Bayesian Methods*. Kluwer Academic, 1989.

